# Optimal Response Initiation:
# Why Recent Experience Matters

**Matt Jones**
Dept. of Psychology &
Institute of Cognitive Science
University of Colorado
mcj@colorado.edu

**Michael C. Mozer**
Dept. of Computer Science &
Institute of Cognitive Science
University of Colorado
mozer@colorado.edu

**Sachiko Kinoshita**
MACCS &
Dept. of Psychology
Macquarie University
skinoshi@maccs.mq.edu.au

## Abstract

In most cognitive and motor tasks, speed-accuracy tradeoffs are observed: Individuals can respond slowly and accurately, or quickly yet be prone to errors. Control mechanisms governing the initiation of behavioral responses are sensitive not only to task instructions and the stimulus being processed, but also to the recent stimulus history. When stimuli can be characterized on an easy-hard dimension (e.g., word frequency in a naming task), items preceded by easy trials are responded to more quickly, and with more errors, than items preceded by hard trials. We propose a rationally motivated mathematical model of this sequential adaptation of control, based on a diffusion model of the decision process in which difficulty corresponds to the drift rate for the correct response. The model assumes that responding is based on the posterior distribution over which response is correct, conditioned on the accumulated evidence. We derive this posterior as a function of the drift rate, and show that higher estimates of the drift rate lead to (normatively) faster responding. Trial-by-trial tracking of difficulty thus leads to sequential effects in speed and accuracy. Simulations show the model explains a variety of phenomena in human speeded decision making. We argue this passive statistical mechanism provides a more elegant and parsimonious account than extant theories based on elaborate control structures.

## 1   Introduction

Consider the task of naming the sum of two numbers, e.g., 14+8. Given sufficient time, individuals will presumably produce the correct answer. However, under speed pressure, mistakes occur. In most cognitive and motor tasks, speed-accuracy tradeoffs are observed: Individuals can respond accurately but slowly, or quickly but be prone to errors. Speed-accuracy tradeoffs are due to the fact that evidence supporting the correct response accumulates gradually over time (Rabbitt & Vyas, 1970; Gold & Shadlen, 2002). Responses initiated earlier in time will be based on lower-quality information, and hence less likely to be correct.

On what basis do motor systems make the decision to initiate a response? Recent theories have cast response initiation in terms of optimality (Bogacz et al., 2006), where optimality might be defined as maximizing reward per unit time, or minimizing a linear combination of latency and error rate. Although optimality might be defined in various ways, all definitions require an estimate of the probability that each candidate response will be correct. We argue that this estimate in turn requires knowledge of the task *difficulty*, or specifically, the rate at which evidence supporting the correct response accumulates over time. If a task is performed repeatedly, task difficulty can be estimated over a series of trials, suggesting that optimal decision processes should show *sequential effects*, in which performance on one trial depends on the difficulty of recent trials. We describe an experimental paradigm that offers behavioral evidence of sequential effects in response initiation.

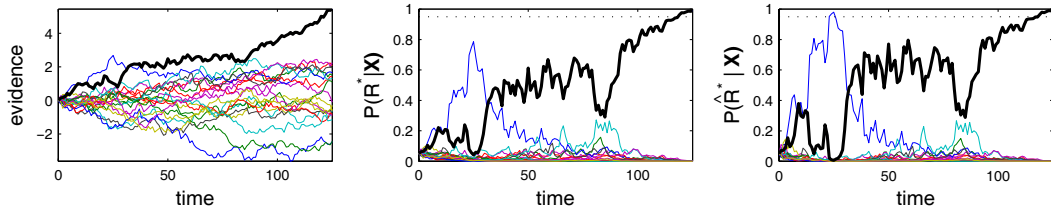

Figure 1: An illustration of the MDM. Left panel: evidence accumulation for a 20-AFC task as a function of time, with $\mu_{R^*} = .04$, $\mu_{i \neq R^*} = 0$, $\sigma = .15$. Middle panel: the posterior over responses, $P(R^*|\mathbf{X})$, with $a = .04$ and $b = 0$, based on the diffusion trace in the left panel. Right panel: the posterior over responses, $P(\hat{R}^*|\mathbf{X})$, assuming $\hat{a} = .07$ and $\hat{b} = .02$ for the same diffusion trace.

We summarize key phenomena from this paradigm, and show that these phenomena are predicted by a model of response initiation. Our work achieves two goals: (1) offering a better understanding of and a computational characterization of control processes involved in response initiation, and (2) offering a rational basis for sequential effects in simple stimulus-response tasks.

## 2 Models of Decision Making

Neurophysiological and psychological data (e.g., Gold & Shadlen, 2002; Ratcliff, Cherian, & Segraves, 2003) have provided converging evidence for a theory of cortical decision making, known as the *diffusion decision model* or *DDM* (see recent review by Ratcliff & McKoon, 2007). The DDM is formulated for two-alternative forced choice (2AFC) decisions. A noisy neural integrator accumulates evidence over time; positive evidence supports one response, negative evidence the other. The model's dynamics are represented by a differential equation, $dx = \mu dt + w$, where $x$ is the accumulated evidence over time $t$, $\mu$ is the relative rate of evidence supporting one response over the other (positive or negative, depending on the balance of evidence), and $w$ is white noise, $w \sim \mathcal{N}(0, \sigma^2 dt)$. The variables $\mu$ and $\sigma$ are called the *drift* and *diffusion* rates. A response is initiated when the accumulated evidence reaches a positive or negative threshold, i.e., $x > \theta^+$ or $x < \theta^-$. The DDM implements the optimal decision strategy under various criteria of optimality (Bogacz et al., 2006).

Tasks involving $n$ alternative responses ($n$AFC) can be modeled by generalizing the DDM to have one integrator per possible response (Bogacz & Gurney, 2007; Vickers, 1970). We refer to this generalized class of models as *multiresponse diffusion models* or *MDM*. Consider one example of an $n$AFC task: naming the color of a visually presented color patch. The visual system produces a trickle of evidence for the correct or *target* response, $R^*$. This evidence supports the target response via a positive drift rate, $\mu_{R^*}$, whereas the drift rates of the other possible color names, $\{\mu_i \mid i \neq R^*\}$, are zero. (We assume no similarity among the stimuli, e.g., an aqua patch provides no evidence for the response 'blue', although our model could be extended in this way.) The left panel of Figure 1 illustrates typical dynamics of the MDM. The abcissa represents processing time relative to the onset of the color patch, and each curve represents one integrator (color name).

### 2.1 A Decision Rule for the Multiresponse Diffusion Model

Although the DDM decision rule is optimal, no unique optimal decision rule exists for the multiple-response case (Bogacz & Gurney, 2007; Dragelin et al, 1999). Rules based on an evidence criterion—analogous to the DDM decision rule—turn out to be inadequate. Instead, candidate rules are based on the posterior probability that a particular response is correct given the observed evidence up to the current time, $P(R^* = r|\mathbf{X})$. In our notation, $R^*$ is the random variable denoting the target response, $r$ is a candidate response among the $n$ alternatives, and $\mathbf{X} = \{x_i(j\tau) \mid i = 1...n, j = 0...\frac{T}{\tau}\}$ is a collection of discrete samples of the multivariate diffusion process observed up to the current time $T$. The simulations reported here use a decision rule that initiates responding when the accuracy of the response is above a threshold, $\theta$:

$$\text{If } \exists r \text{ such that } P(R^* = r|\mathbf{X}) \geq \theta, \text{ then initiate response } r. \tag{1}$$

This rule has been shown to minimize decision latency in the limit of $\theta \to 1$ (Dragelin et al., 1999). However, our model's predictions are not tied to this particular rule. We emphasize that any sensible rule requires estimation of $P(R^* = r|\mathbf{X})$, and we focus on how the phenomena explained by our model derive from the properties of this posterior distribution.

Baum and Veeravalli (1994; see also Bogacz & Gurney, 2007) derive $P(R^* = r|\mathbf{X})$ for the case where all nontargets have the same drift rate, $\mu_{\mathrm{nontgt}}$, the target has drift rate $\mu_{\mathrm{tgt}}$, and $\mu_{\mathrm{nontgt}}$, $\mu_{\mathrm{tgt}}$, and $\sigma$ are known. (We introduce the $\mu_{\mathrm{tgt}}$ and $\mu_{\mathrm{nontgt}}$ notation to refer to these drift rates even in the absence of information about $R^*$.) We extend the Baum and Veeravalli result to the case where $\mu_{\mathrm{tgt}}$ is an unknown random variable that must be estimated by the observer. The diffusion rate of a random walk, $\sigma^2$, can be determined with arbitrary precision from a single observed trajectory, but the drift rate cannot (see Supplementary Material – available at http://matt.colorado.edu/papers.htm). Therefore, estimating statistics of $\mu_{\mathrm{tgt}}$ is critical to achieving optimal performance.

Given a sequence of discrete observations from a diffusion process, $\mathbf{x} = \{x(j\tau) \mid j = 0...\frac{T}{\tau}\}$, we can use the independence of increments to a diffusion process with known drift and diffusion rates, $x(t_2) - x(t_1) \sim \mathcal{N}\left((t_2 - t_1)\mu, (t_2 - t_1)\sigma^2\right)$, to calculate the likelihood of $\mathbf{x}$:

$$P(\mathbf{x}|\mu, \sigma) \propto \exp\left[(\Delta x(T)\mu - \mu^2 T/2)/\sigma^2\right], \tag{2}$$

where $\Delta x(T) = x(T) - x(0)$ is a sufficient statistic for estimating $\mu$.

Consider the case where the drift rate of the target is a random variable, $\mu_{\mathrm{tgt}} \sim \mathcal{N}(a, b^2)$, and the drift rate of all nontargets, $\mu_{\mathrm{nontgt}}$, is zero. Using Equation 2 and integrating out $\mu_{\mathrm{tgt}}$, the posterior over response alternatives can be determined (see Supplementary Material):

$$P(R^* = r|\mathbf{X}, a, b) \propto \exp\left[\frac{b^2\Delta x_r(T)^2 + 2a\sigma^2\Delta x_r(T)}{2\sigma^2(\sigma^2 + Tb^2)}\right]. \tag{3}$$

The middle panel of Figure 1 shows $P(R^*|\mathbf{X}, a, b)$, as a function of processing time for the diffusion trace in the left panel, when the true drift rate is known ($a = \mu_{\mathrm{tgt}}$ and $b = 0$).

## 2.2 Estimating Drift

To recap, we have argued that optimal response initiation in $n$AFC tasks requires calculation of the posterior response distribution, which in turn depends on assumptions about the drift rate of the target response. We proposed a decision rule based on a probabilisitic framework (Equations 1 and 3) that permits uncertainty in the drift rate, but requires a characterization of the prior distribution of this variable.

We assume that the parameters of this distribution, $a$ and $b$, are unknown. Consequently, the observer cannot compute $P(R^*|\mathbf{X})$, but must use an approximation, $P(\hat{R}^*|\mathbf{X})$, based on estimates $\hat{a}$ and $\hat{b}$. When $\mu_{\mathrm{tgt}}$ is not representative of the assumed distribution $\mathcal{N}(\hat{a}, \hat{b}^2)$, performance of the model will be impaired, as illustrated by a comparison of the center and right panels of Figure 1. In the center panel, $\mu_{\mathrm{tgt}} = .04$ is known; in the right panel, $\mu_{\mathrm{tgt}}$ is not representative of the assumed distribution. The consequence of this mismatch is that—for the criterion indicated by the dashed horizontal line—the model chooses the wrong response.

We turn now to the estimation of the model's drift distribution parameters, $\hat{a}$ and $\hat{b}$. Consider a sequence of trials, $k = 1...K$, in which the same decision task is performed with different stimuli, and the drift rate of the target response on trial $k$ is $\mu(k)$. Following each trial, the drift rate can also be estimated: $\hat{\mu}_{\mathrm{tgt}}(k) = \Delta x_{R^*}(T_k)/T_k$, where $T_k$ is the time taken to respond on trial $k$. If the task environment changes slowly, the drift rates over trials will be autocorrelated, and the drift distribution parameters on trial $k$ can be estimated from past trial history, $\{\hat{\mu}_{\mathrm{tgt}}(1)...\hat{\mu}_{\mathrm{tgt}}(k - 1)\}$. The weighting of past history should be based on the strength of the autocorrelation. Using maximum likelihood estimation of $a$ and $b$ with an exponential weighting on past history, one obtains

$$\hat{a}(k) = v_1(k)/v_0(k), \quad \text{and} \quad \hat{b}(k) = [v_2(k)/v_0(k) - \hat{a}(k)^2]^{0.5} \tag{4}$$

where $k$ is an index over trials, and the $\{v_i(k)\}$ are moment statistics of the drift disribution, updated following each trial using an exponential weighting constant, $\lambda \in [0, 1]$:

$$v_i(k) = \lambda v_i(k - 1) + \hat{\mu}_{\mathrm{tgt}}(k - 1)^i. \tag{5}$$

This update rule is an efficient approximation to full hierarchical Bayesian inference of $a$ and $b$. When combined with Equations 1 and 3 it determines the model's response on the current trial.

# 3 The Blocking Effect

The optimal decision framework we have proposed naturally leads to the prediction that performance on the current trial is influenced by drift rates observed on recent trials. Because drift rates determine the signal-to-noise ratio of the diffusion process, they reflect the *difficulty* of the task at hand. Thus, the framework predicts that an optimal decision maker should show sequential effects based on recent trial difficulty. We now turn to behavioral data consistent with this prediction.

In any behavioral task, some items are intrinsically easier than others, e.g., 10+3 is easier than 5+8, whether due to practice or the number of cognitive operations required to determine the sum. By definition, individuals have faster response times (RTs) and lower error rates to easy items. However, the RTs and error rates are modulated by the composition of a trial block. Consider an experimental paradigm consisting of three blocks: just easy items (*pure easy*), just hard items (*pure hard*), and a mixture of both in random order (*mixed*). When presented in a mixed block, easy items slow down relative to a pure block and hard items speed up. This phenomenon, known as the *blocking effect* (not to be confused with blocking in associative learning), suggests that the response-initiation processes use information not only from the current stimulus, but also from the stimulus environment in which it is operating. Table 1 shows a typical blocking result for a word-reading task, where word frequency is used to manipulate difficulty. We summarize the central, robust phenomena of the blocking-effect literature (e.g., Kiger & Glass, 1981; Lupker, Brown & Columbo, 1997; Lupker, Kinoshita, Coltheart, & Taylor, 2000; Taylor & Lupker, 2001).

*P1.* Blocking effects occur across diverse paradigms, including naming, arithmetic verification and calculation, target search, and lexical decision. They are obtained when stimulus or response characteristics alternate from trial to trial. Thus, the blocking effect is not associated with a specific stimulus or response pathway, but rather is a general phenomenon of response initiation.

*P2.* A signature of the effect concerns the relative magnitudes of easy-item slowdown and hard-item speedup. Typically, slowdown and speedup are of equal magnitude. Significantly more speedup than slowdown is never observed. However, in some paradigms (e.g., lexical decision, priming) significantly more slowdown than speedup can be observed.

*P3.* The RT difference bewteen easy and hard items does not fully disappear in mixed blocks. Thus, RT depends on both the stimulus type and the composition of the block.

*P4.* Speed-accuracy tradeoffs are observed: A drop in error rate accompanies easy-item slowdown, and a rise in error rate accompanies hard-item speedup.

*P5.* The effects of stimulus history are local, i.e., the variability in RT on trial $k$ due to trial $k - l$ decreases rapidly with $l$. Dependencies for $l > 2$ are not statistically reliable (Taylor & Lupker, 2001), although the experiments may not have had sufficient power to detect weak dependencies.

*P6.* Overt responses are necessary for obtaining blocking effects, but overt errors are not.

# 4 Explanations for the Blocking Effect

The blocking effect demonstrates that the response time depends not only on information accruing from the current stimulus, but also on recent stimuli in the trial history. Therefore, any explanation of the blocking effect must specify the manner by which response initiation processes are sensitive to the composition of a block. Various mechanisms of control adaptation have been proposed.

*Domain-specific mechanisms.* Many of the proposed mechanisms are domain-specific. For example, Rastle and Coltheart (1999) describe a model with two routes to naming, one lexical and one non-lexical, and posit that the composition of a block affects the emphasis that is placed on the output of one route versus the other. Because of the ubiquity of blocking effects across tasks, domain-specific

Table 1: RTs and Error Rates for Blocking study of Lupker, Brown, & Columbo (1997, Expt. 3)

|  | Pure Block | Mixed Block | Difference |
|---|---|---|---|
| Easy | 488 ms (3.6%) | 513 ms (1.8%) | +25 ms (-1.8%) |
| Hard | 583 ms (12.0%) | 559 ms (12.2%) | -24 ms (+0.2%) |

accounts are not compelling. Parsimony is achieved only if the adaptation mechanism is localized to a stage of response initiation common across stimulus-response tasks.

*Rate of convergence.* Kello and Plaut (2003) have proposed that control processes adjust a gain parameter on units in a dynamical connectionist model. Increasing the gain results in more rapid convergence, but also a higher error rate. Simulations of this model have explained the basic blocking effect, but not the complete set of phenomena we listed previously. Of greater concern is the fact that the model predicts the time taken to utter the response (when the response mode is verbal) decreases with increased speed pressure, which does not appear to be true (Damian, 2003).

*Evidence criterion.* A candidate mechanism with intuitive appeal is the trial-to-trial adjustment of an evidence criterion in the MDM, such that the easier the previous trials are, the lower the criterion is set. This strategy results in the lowest criterion in a pure-easy block, intermediate in a mixed block, and highest in a pure-hard block. Because a higher criterion produces slower RTs and lower error rates, this leads to slowdown of easy items and speedup of hard items in a mixed block. Nonetheless, there are four reasons for being skeptical about an account of the blocking effect based on adjustment of an evidence criterion. (1) From a purely computational perspective, the optimality—or even the behavioral robustness—of an MDM with an evidence criterion has not been established. (2) Taylor and Lupker (2001) illustrate that adaptation of an evidence criterion can—at least in some models—yield incorrect predictions concerning the blocking effect. (3) Strayer and Kramer (1994) attempted to model the blocking effect for a 2AFC task using an adaptive response criterion in the DDM. Their account fit data, but had a critical shortcoming: They needed to allow different criteria for easy and hard items in a mixed block, which makes no sense because the trial type was not known in advance, and setting differential criteria depends on knowing the trial type. (4) On logical grounds, the relative importance of speed versus accuracy should be determined by task instructions and payoffs. Item difficulty is an independent and unrelated factor. Consistent with this logical argument is the finding that manipulating instructions to emphasize speed versus accuracy does not produce the same pattern of effects as altering the composition of a block (Dorfman & Glanzer, 1988).

## 5 Our Account: Sequential Estimation of Task Difficulty

Having argued that existing accounts of the blocking effect are inadequate, we return to our analysis of $n$AFC tasks, and show that it provides a parsimonious account of blocking effects. Our account is premised on the assumption that response initiation processes are in some sense optimal. Regardless of the specific optimality criterion, optimal response initiation requires an estimate of accuracy, specifically, the probability that a response will be correct conditioned on the evidence accumulated thus far, $P(R^* = r|\mathbf{X})$. As we argue above, estimation of this probability requires knowledge of the difficulty (drift) of the correct response, and recent trial history can provide this information.

The response posterior, $P(R^* = r|\mathbf{X})$, under our generative model of the task environment (Equation 3) predicts a blocking effect. To see this clearly, consider the special case where uncertainty in $\mu_{\text{tgt}}$ is negligible, i.e., $b \to 0$, which simplifies Equation 3 to $P(R^* = r|\mathbf{X}) \propto \exp\left[a\Delta x_r(T)/\sigma^2\right]$. This expression is a Gibbs distribution with temperature $\sigma^2/a$. As the temperature is lowered, the entropy drops, and the probabilities become more extreme. Thus, larger values of $a$ lead to faster responses, because the greater expected signal-to-noise ratio makes evidence more reliable. How does this fact relate to the blocking effect? Easy items have, by definition, a higher mean drift than hard items; therefore, the estimated drift in the easy condition will be greater than in the hard condition, $E[\hat{a}_E] > E[\hat{a}_H]$. Any learning rule for $\hat{a}$ based on recent history will yield an estimated drift in the mixed condition between those of the easy and hard conditions, i.e., $E[\hat{a}_E] > E[\hat{a}_M] > E[\hat{a}_H]$. With response times related to $\hat{a}$, an easy item will slow down in the mixed condition relative to the pure, and a hard item will speed up.

Although we could fit behavioral data (e.g., Table 1) quantitatively, such fits add no support for the model beyond a qualitative fit. The reason lies in the mapping of model decision times to human response latencies. An affine transform must be allowed, scaling time in the model to real-world time, and also allowing for a fixed-duration stage of perceptual processing. A blocking effect of any magnitude in the model could therefore be transformed to fit any pattern of data that had the right qualitative features. We thus focus on qualitative performance of the model.

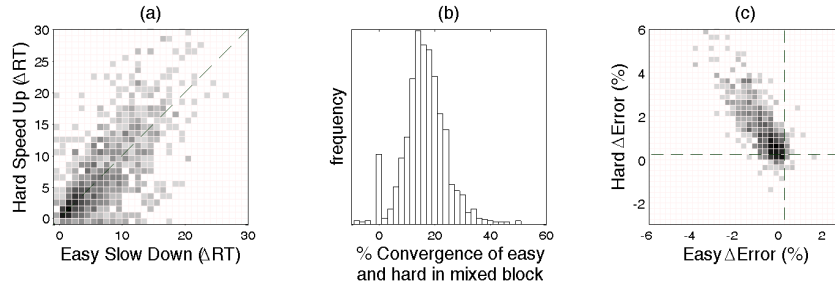

Figure 2: Simulation of the blocking paradigm with random parameter settings. (a) Scatterplot of hard speedup vs. easy slowdown, where coloring of a cell reflects the log(frequency) with which a given simulation outcome is obtained. (b) Histogram of percentage reduction in the difference between easy and hard RTs as a result of intermixing. (c) Scatterplot of change in error rate between pure and mixed conditions for easy and hard items.

The model has four internal parameters: $\sigma$ (diffusion rate), $\lambda$ (history decay), $\theta$ (accuracy criterion), and $n$ (number of response alternatives). In addition, to simulate the blocking effect, we must specify the true drift distributions for easy and hard items, i.e., $a_E$, $b_E$, $a_H$, and $b_H$. (We might also allow for nonzero drift rates for some or all of the distractor responses.) To explore the robustness of the model, we performed 1200 replications of a blocking simulation, each with randomly drawn values for the eight free parameters. Parameters were drawn as follows: $\sigma \sim U(.05, .25)$, $\lambda \sim 1 - 1/(1 + U(1, 20))$ (these values are uniform in the half-life of the exponential memory decay), $n \sim \lfloor U(2, 100) \rfloor$, $\theta \sim U(.95, .995)$, $a_H \sim U(.01, .05)$, $a_E \sim a_H + U(.002, .02)$, $b_H \sim (a_E - a_H)/U(3, 10)$, and $b_E = b_H$. Each replication involved simulating three conditions: pure easy, pure hard, and mixed. The pure conditions were run for 5000 trials and the mixed condition for 10000 trials. Each condition began with an additional 25 practice trials which were discarded from our analysis but were useful to eliminate the effects of initialization of $\hat{a}$ and $\hat{b}$. The model parameters were not adapted following error trials. For each replication and each condition, the median response time (RT) and mean error rate were computed. We discarded from our analysis simulations in which the error rates were grossly unlike those obtained in experimental studies, specifically, where the mean error rate in any condition was above 20%, and where the error rates for easy and hard items differed by more than a factor of 10.

Figure 2a shows a scatterplot comparing the speedup of hard items (from pure to mixed conditions) to the slowdown of easy items. Units are in simulation time steps. The dashed diagonal line indicates speedup comparable in magnitude to slowdown. Much of the scatter is due to sampling noise in the median RTs. The model obtains a remarkably symmetric effect: 41% of replications yield speedup > slowdown, 40% yield slowdown > speedup, and the remaining 19% yield exactly equal sized effects. The slope of the regression line through the origin is 0.97. Thus, the model shows a key signature of the behavioral data—symmetric blocking effects (Phenomenon P2).

Figure 2b shows a histogram of the percentage reduction in the difference between easy and hard RTs as a result of intermixing. This percentage is 100 if easy RTs slow down and hard RTs speed up to become equal; the percentage is 0 if there is no slowdown of easy RTs or speedup of hard RTs. The simulation runs show a 10–30% reduction as a result of the blocking manipulation. This percentage is unaffected by the affine transformation required to convert simulation RTs to human RTs, and is thus directly comparable. Behavioral studies (e.g., Table 1) typically show 20–60% effects. Thus, the model—with random parameter settings—tends to underpredict human results. Nonetheless, the model shows the key property that easy RTs are still faster than hard RTs in the mixed condition (Phenomenon P3).

Figure 2c shows a scatterplot of the change in error rate for easy items (from pure to mixed conditions) versus change in error rate for hard items. Consistent with the behavioral data (Phenomenon P4), a speed-accuracy trade off is observed: When easy items slow down in the mixed versus pure conditions, error rates drop; when hard items speed up, error rates rise. This trade off is expected, because block composition affects only the stopping point of the model and not the model dynamics. Thus, any speedup should yield a higher error rate, and vice versa. Interestingly, the accuracy

Figure 3: Human (black) and simulation (white) RTs for easy and hard items in a mixed block, conditional on the 0, 1, and 2 previous items (Taylor & Lupker, 2001). Last letter in the trial sequence indicates the current trial and trial order is left to right.

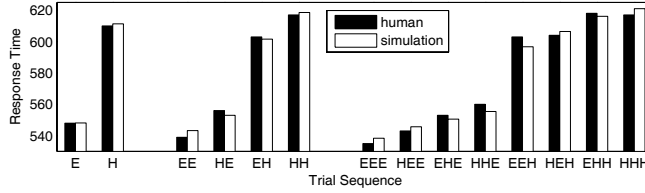

criterion is fixed across conditions in the model; the differences in error rates arise because of a mismatch between the parameters $a$ and $b$ used to generate trials, and the parameters $\hat{a}$ and $\hat{b}$ estimated from the trial sequence. Thus, although the criterion does not change across conditions, and the criterion is expressed in terms of accuracy (Equation 1), the block composition nonetheless affects the speed-accuracy trade off.

Although the blocking effect is typically characterized by comparing performance of an item type across blocks, sequential effects within a block have also been examined. Taylor and Lupker (2001, Experiment 1) instructed participants to name high-frequency words (easy items) and nonwords (hard items). Focusing on the mixed block, Taylor and Lupker analyzed RTs conditional on the context—the 0, 1, and 2 preceding items. The black bars in Figure 3 show the RTs conditional on the context. Trial $k$ is most influenced by trial $k-1$, but trial $k-2$ modulates RTs as well. This decreasing influence of previous trials (Phenomenon P5) is well characterized by the model via the exponential-decay parameter, $\lambda$ (Equation 5). To model the Taylor and Lupker data, we ran a simulation with generic parameters which were not tuned to the data: $a_E = .05$, $a_H = .04$, $b_E = b_H = .002$, $\sigma = .15$, $\theta = .99$, $\lambda = .5$, and $n = 5$. We then scaled simulation RTs to human RTs with an affine transform whose two free parameters were fit to the data. The result, shown by the white bars in Figure 3, captures the important properties of the data.

We have addressed all of the key phenomena of the blocking effect except two. Phenomenon P1 concerns the fact that the effect occurs across a variety of tasks and difficulty manipulations. The ubiquity of the effect is completely consistent with our focus on general mechanisms of response initiation. The model does not make any claims about the specific domain or the cause of variation in drift rates. Phenomenon P6 states that overt responses are required to obtain the blocking effect. Although the model cannot lay claims to distinctions between overt and covert responses, it does require that a drift estimate, $\hat{\mu}_{\text{tgt}}$, be obtained on each trial in order to adjust $\hat{a}$ and $\hat{b}$, which leads to blocking effects. In turn, $\hat{\mu}_{\text{tgt}}$ is determined at the point in the diffusion process when a response would be initiated. Thus, the model claims that selecting a response on trial $k$ is key to influencing performance on trial $k+1$.

## 6 Conclusions

We have argued that optimal response initiation in speeded choice tasks requires advance knowledge about the difficulty of the current decision. Difficulty corresponds to the expected rate of evidence accumulation for the target response relative to distractors. When difficulty is high, the signal-to-noise ratio of the evidence-accumulation process is low, and a rational observer will wait for more evidence before initiating a response.

Our model assumes that difficulty in the current task environment is estimated from the difficulty of recent trials, under an assumption of temporal autocorrelation. This is consistent with the empirically observed blocking effect, whereby responses are slower to easy items and faster to hard items when those items are interleaved, compared to when item types are presented in separate blocks. According to our model, mixed blocks induce estimates of local difficulty that are intermediate between those in pure easy and pure hard blocks. The resultant overestimation of difficulty for easy items leads to increased decision times, while an opposite effect occurs for hard items.

We formalize these ideas in a multiresponse diffusion model of decision making. Evidence for each response accrues in a random walk, with positive drift rate $\mu_{\text{tgt}}$ for the correct response and zero drift for distractors. Analytical derivations show that conversion of evidence to a posterior distribution

over responses depends on $\mu_{\text{tgt}}$, which acts as an inverse temperature in a Gibbs distribution. When this parameter is uncertain, with a prior estimated from recent context, error in the estimate leads to systematic bias in the response time. Underestimation of the drift rate, as with easy trials in a mixed block, leads to damping of the computed posterior and response slowdown. Overestimation, as with hard trials in a mixed block, leads to exaggeration of the posterior and response speedup.

The model successfully explains the full range of phenomena associated with the blocking effect, including the effects on both RTs and errors, the patterns of slowdown of easy items and speedup of hard items, and the detailed sequential effects of recent trials. Moreover, the model is robust to parameter settings, as our random-replication simulation shows. The model is robust in other respects as well: Its qualitative behavior does not depend on the number of response alternatives (we have tried up to 1000), the decision rule (we have also tried a criterion based on the posterior ratio between the most and next most probable responses), the estimation algorithm for $\hat{a}$ and $\hat{b}$ (we have also tried a Kalman filter), and violations of assumptions of the generative model (e.g., nonzero drift rates for some of the distractors, reflecting the similarity structure of perceptual representations).

The tradeoff between speed and accuracy in decision making is a paradigmatic problem of cognitive control. Theories in cognitive science often hand the problem of control to a homunculus. When control processes are specified, they generally involve explicit, active, and sophisticated mechanisms (e.g., conflict detection; A.D. Jones et al., 2002). Our model achieves sequential adaptation of control via a statistical mechanism that is passive and in a sense dumb; it essentially reestimates the statistical structure of the environment by updating an expectation of task difficulty. Our belief is that many aspects of cognitive control can be explained away by such passive statistical mechanisms, eventually eliminating the homunculus from cognitive science.

### Acknowledgments

This research was supported by NSF grants BCS-0339103, BCS-720375, SBE-0518699, and SBE-0542013, and ARC Discovery Grant DP0556805. We thank the students in CSCI7222/CSCI4830/PSYC7782 for interesting discussions that led to this work.

### References

Baum, C. W., & Veeravalli, V. (1994). A sequential procedure for multi-hypothesis testing. *IEEE Trans. Inf. Theory*, *40*, 1994–2007.

Bogacz, R, Brown, E, Moehlis, J, Holmes, P & Cohen JD (2006). The physics of optimal decision making: A formal analysis of models of performance in two-alternative forced choice tasks. *Psych. Rev.*, *113*, 700–765.

Bogacz, R. & Gurney, K. (2007). The basal ganglia and cortex implement optimal decision making between alternative actions. *Neural Computation*, *19*, 442-477.

Damian, M. F. (2003). Articulatory duration in single word speech production. *JEP: LMC*, *29*, 416–431.

Dorfman, D., & Glanzer, M. (1988). List composition effects in lexical decision and recognition memory. *J. Mem. & Lang.*, *27*, 633–648.

Gold, J.I., & Shadlen, M.N. (2002). Banburismus and the brain: Decoding the relationship between sensory stimuli, decisions and reward. *Neuron*, *36*, 299–308.

Jones, A. D., Cho, R. Y., Nystrom, L. E., Cohen, J. D., & Braver, T. S. (2002). A computational model of anterior cingulate function in speeded response tasks: Effects of frequency, sequence, and conflict. *Cogn., Aff., & Beh. Neuro.*, *2*, 300–317.

Kello, C. T. & Plaut, D. C. (2003). Strategic control over rate of processing in word reading: A computational investigation. *J. Mem. & Lang.*, *48*, 207–232.

Kiger, J. I., & Glass, A. L. (1981). Context effects in sentence verification. *JEP:HPP*, *7*, 688–700.

Lupker, S. J., Brown, P., & Colombo, L. (1997). Strategic control in a naming task: Changing routes or changing deadlines? *JEP:LMC*, *23*, 570–590.

Rabbitt, PMA, & Vyas, SM (1970). An elementary preliminary taxonomy for some errors in laboratory choice RT tasks. Acta Psych., 33, 56-76.

Rastle, K., & Coltheart, M. (1999). Serial and strategic effects in reading aloud. *JEP:HPP*, *25*, 482–503.

Ratcliff, R., & McKoon, G. (2007). The diffusion decision model: Theory and data for two-choice decision tasks. *Neural Computation*, *20*, 873–922.

Ratcliff, R., Cherian, A., & Segraves, M. (2003) A comparison of macaque behavior and superior colliculus neuronal activity to predictions from models of two-choice decisions. *J. Neurophys.*, *90*, 1392–1407.

Taylor, T. E., & Lupker, S. J. (2001). Sequential effects in naming: A time-criterion account. *Journal of Experimental Psychology: Learning, Memory, and Cognition*, *27*, 117–138.

